# Learning from Neighboring Strokes: Combining Appearance and Context for Multi-Domain Sketch Recognition

**Tom Y. Ouyang**     **Randall Davis**
Computer Science and Artificial Intelligence Laboratory
Massachusetts Institute of Technology
Cambridge, MA 02139 USA
`{ouyang,davis}@csail.mit.edu`

## Abstract

We propose a new sketch recognition framework that combines a rich representation of low level visual appearance with a graphical model for capturing high level relationships between symbols. This joint model of appearance and context allows our framework to be less sensitive to noise and drawing variations, improving accuracy and robustness. The result is a recognizer that is better able to handle the wide range of drawing styles found in messy freehand sketches. We evaluate our work on two real-world domains, molecular diagrams and electrical circuit diagrams, and show that our combined approach significantly improves recognition performance.

## 1 Introduction

Sketches are everywhere. From flow charts to chemical structures to electrical circuits, people use them every day to communicate information across many different domains. They are also be an important part of the early design process, helping us explore rough ideas and solutions in an informal environment. However, despite their ubiquity, there is still a large gap between how people naturally interact with sketches and how computers can interpret them today. Current authoring programs like ChemDraw (for chemical structures) and Visio (for general diagrams) still rely on the traditional point-click-drag style of interaction. While popular, they simply do not provide the ease of use, naturalness, or speed of drawing on paper.

We propose a new framework for sketch recognition that combines a rich representation of low level visual appearance with a probabilistic model for capturing higher level relationships. By "visual appearance" we mean an image-based representation that preserves the pictoral nature of the ink. By "higher level relationships" we mean the spatial relationships between different symbols. Our combined approach uses a graphical model that classifies each symbol jointly with its context, allowing neighboring interpretations to influence each other. This makes our method less sensitive to noise and drawing variations, significantly improving robustness and accuracy. The result is a recognizer that is better able to handle the range of drawing styles found in messy freehand sketches.

Current work in sketch recognition can, very broadly speaking, be separated into two groups. The first group focuses on the relationships between geometric primitives like lines, arcs, and curves, specifying them either manually [1, 4, 5] or learning them from labeled data [16, 20]. Recognition is then posed as a constraint satisfaction problem, as in [4, 5], or as an inference problem on a graphical model, as in [1, 16, 17, 20]. However, in many real-world sketches, it is difficult to extract these primitives reliably. Circles may not always be round, line segments may not be straight, and stroke artifacts like pen-drag (not lifting the pen between strokes), over-tracing (drawing over a

previously drawn stroke), and stray ink may introduce false primitives that lead to poor recognition. In addition, recognizers that rely on extracted primitives often discard potentially useful information contained in the appearance of the original strokes.

The second group of related work focuses on the visual appearance of shapes and symbols. These include parts-based methods [9, 18], which learn a set of discriminative parts or patches for each symbol class, and template-based methods [7, 11], which compare the input symbol to a library of learned prototypes. The main advantage of vision-based approaches is their robustness to many of the drawing variations commonly found in real-world sketches, including artifacts like over-tracing and pen drag. However, these methods do not model the spatial relationships between neighboring shapes, relying solely on local appearance to classify a symbol.

In the following sections we describe our approach, which combines both appearance and context. It is divided into three main stages: (1) stroke preprocessing: we decompose strokes (each stroke is defined as the set of points collected from pen-down to pen-up) into smaller segments, (2) symbol detection: we search for potential symbols (candidates) among groups of segments, and (3) candidate selection: we select a final set of detections from these candidates, taking into account their spatial relationships.

## 2 Preprocessing

The first step in our recognition framework is to preprocess the sketch into a set of simple segments, as shown in Figure 1(b). The purpose for this step is twofold. First, like superpixels in computer vision [14], segments are much easier to work with than individual points or pixels; the number of points can be large even in moderate-sized sketches, making optimization intractable. Second, in the domains we evaluated, the boundaries between segments effectively preserve the boundaries between symbols. This is not the case when working with the strokes directly, so preprocessing allows us to handle strokes that contain more than one symbol (e.g., when a wire and resistor are drawn together without lifting the pen).

Our preprocessing algorithm divides strokes into segments by splitting them at their corner points. Previous approaches to corner detection focused primarily on local pen speed and curvature [15], but these measures are not always reliable in messy real-world sketches. Our corner detection algorithm, on the other hand, tries to find the set of vertices that best approximates the original stroke as a whole. It repeatedly discards the vertex $v_i$ that contributes the least to the quality of fit measure $q$, which we define as:

$$q(v_i) = (\mathrm{MSE}(\boldsymbol{v} \setminus v_i, \boldsymbol{s}) - \mathrm{MSE}(\boldsymbol{v}, \boldsymbol{s})) * \mathrm{curvature}(v_i) \tag{1}$$

where $\boldsymbol{s}$ is the set of points in the original stroke, $\boldsymbol{v}$ is the current set of vertices remaining in the line segment approximation, $\mathrm{curvature}(v_i)$ is a measure of the local stroke curvature[1], and $(\mathrm{MSE}(\boldsymbol{v} \setminus v_i, \boldsymbol{s}) - \mathrm{MSE}(\boldsymbol{v}, \boldsymbol{s}))$ is the increase in mean squared error caused by removing vertex $v_i$ from the approximation.

Thus, instead of immediately trying to decide which point *is* a corner, our detector starts by making the simpler decision about which point *is not* a corner. The process ends when $q(v_i)$ is greater than a predefined threshold[2]. At the end of the preprocessing stage, the system records the length of the longest segment $L$ (after excluding the top 5% as outliers). This value is used in subsequent stages as a rough estimate for the overall scale of the sketch.

## 3 Symbol Detection

Our algorithm searches for symbols among groups of segments. Starting with each segment in isolation, we generate successively larger groups by expanding the group to include the next closest segment[3]. This process ends when either the size of the group exceeds $2L$ (a spatial constraint) or

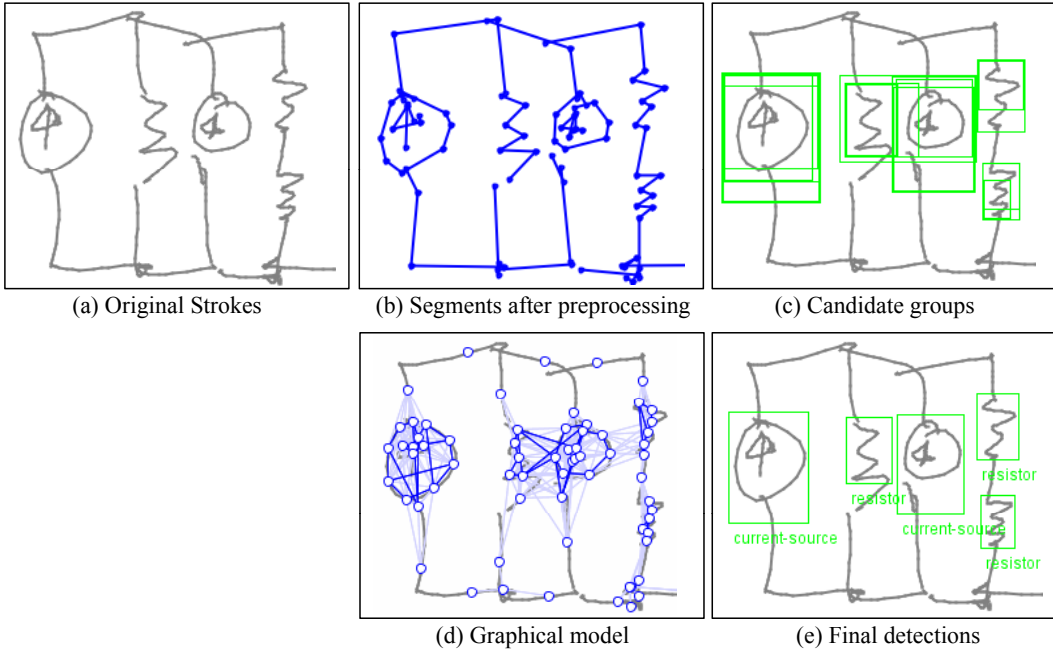

| (a) Original Strokes | (b) Segments after preprocessing | (c) Candidate groups |
| --- | --- | --- |
| (d) Graphical model | | (e) Final detections |

Figure 1: Our recognition framework. (a) An example sketch of a circuit diagram and (b) the segments after preprocessing. (c) A subset of the candidate groups extracted from the sketch (only those with an appearance potential > 0.25 are shown). (d) The resulting graphical model: nodes represent segment labels, dark blue edges represent group overlap potentials, and light blue edges represent context potentials. (e) The final set of symbol detections after running loopy belief propagation.

when the group spans more strokes than the temporal window specified for the domain[4]. Note that we allow temporal gaps in the detection region, so symbols do not need to be drawn with consecutive strokes. An illustration of this process is shown in Figure 1(c).

We classify each candidate group using the symbol recognizer we described in [11], which converts the on-line stroke sequences into a set of low resolution feature images (see Figure 2(a)). This emphasis on visual appearance makes our method less sensitive to stroke level differences like over-tracing and pen drag, improving accuracy and robustness. Since [11] was designed for classifying isolated shapes and not for detecting symbols in messy sketches, we augment its output with five geometric features and a set of local context features:

**stroke count**: The number of strokes in the group.

**segment count**: The number of segments in the group.

**diagonal length**: The diagonal length of the group's bounding box, normalized by $L$.

**group ink density**: The total length of the strokes in the group divided by the diagonal length. This feature is a measure of the group's ink density.

**stroke separation**: Maximum distance between any stroke and its nearest neighbor in the group.

**local context**: A set of four feature images that captures the local context around the group. Each image filters the local appearance at a specific orientation: 0, 45, 90, and 135 degrees. The images are centered at the middle of the group's bounding box and scaled so that each dimension is equal to the group's diagonal length, as shown in Figure 2(b). The initial 12x12 images are smoothed using a Gaussian filter, down-sampled by a factor of 4.

The symbol detector uses a linear SVM [13] to classify each candidate group, labeling it as one of the symbols in the domain or as mis-grouped "clutter". The training data includes both valid symbols and clutter regions. Because the classifier needs to distinguish between more than two classes, we

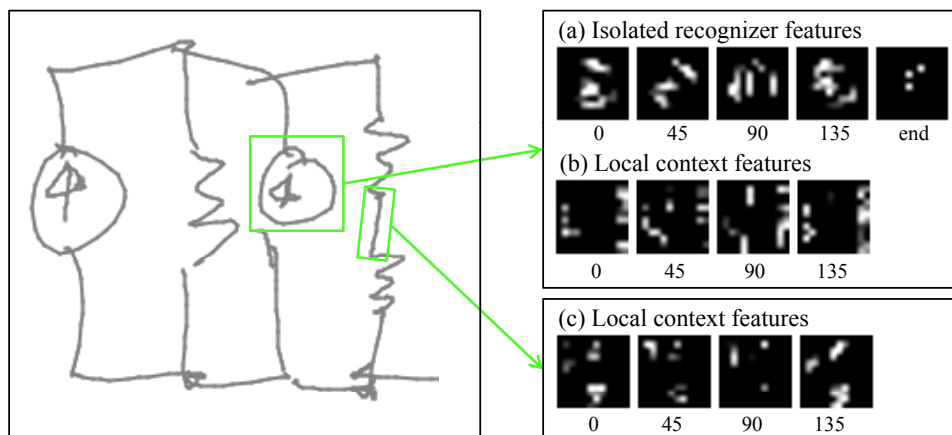

Figure 2: Symbol Detection Features. (a) The set of five 12x12 feature images used by the isolated appearance-based classifier. The first four images encode stroke orientation at 0, 45, 90, and 135 degrees; the fifth captures the locations of stroke endpoints. (b) The set of four local context images for multi-segment symbol. (c) The set of four local context images for single-segment symbols.

use the one-vs-one strategy for combining binary classifiers. Also, to generate probability estimates, we fit a logistic regression model to the outputs of the SVM [12].

Many of the features above are not very useful for groups that contain only one segment. For example, an isolated segment always looks like a straight line, so its visual appearance is not very informative. Thus, we use a different set of features to classify candidates that contain only a single segment: (e.g., wires in circuits and straight bonds in chemistry):

**orientation**: The orientation of the segment, discretized into evenly space bins of size $\pi/4$.

**segment length**: The length of the segment, normalized by $L$.

**segment count**: The total number of segments extracted from the parent stroke.

**segment ink density**: The length of the substroke matching the start and end points of the segment divided by the length of the segment. This is a measure of the segment's curvature and is higher for more curved segments.

**stroke ink density**: The length of the parent stroke divided by the diagonal length of the parent stroke's bounding box.

**local context**: Same as the local context for multi-segment symbols, except these images are centered at the midpoint of the segment, oriented in the same direction as the segment, and scaled so that each dimension is equal to two times the length of the segment. An example is shown in 2(c).

## 4 Improving Recognition using Context

The final task is to select a set of symbol detections from the competing candidate groups. Our candidate selection algorithm has two main objectives. First, it must avoid selecting candidates that conflict with each other because they share one or more segments. Second, it should select candidates that are consistent with each other based on what the system knows about the likely spatial relationships between symbols.

We use an undirected graphical model to encode the relationships between competing candidates. Under our formulation, each segment (node) in the sketch needs to be assigned to one of the candidate groups (labels). Thus, our candidate selection problem becomes a segment labeling problem, where the set of possible labels for a given segment is the set of candidate groups that contain that segment. This allows us to incorporate local appearance, group overlap consistency, and spatial context into a single unified model.

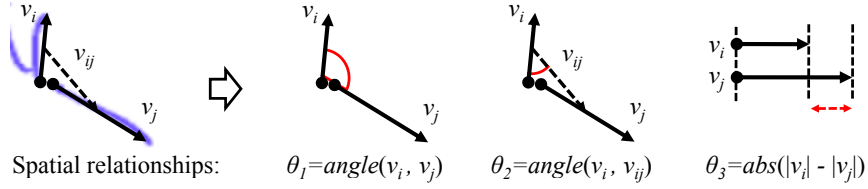

Spatial relationships:    $\theta_1 = angle(v_i, v_j)$    $\theta_2 = angle(v_i, v_{ij})$    $\theta_3 = abs(|v_i| - |v_j|)$

Figure 3: Spatial relationships: The three measurements used to calculate the context potential $\psi_c(c_i, c_j, x_i, x_j)$, where $v_i$ and $v_j$ are vector representing segment $x_i$ and $x_j$ and $v_{ij}$ is a vector from the center of $v_i$ to the center of $v_j$.

The joint probability function over the entire graph is given by:

$$\log P(\mathbf{c}|\mathbf{x}) = \sum_i \overbrace{\psi_a(c_i, \mathbf{x})}^{\text{appearance}} + \sum_{ij} \overbrace{\psi_o(c_i, c_j)}^{\text{overlap}} + \overbrace{\psi_c(c_i, c_j, x_i, x_j)}^{\text{context}} - \log(Z) \qquad (2)$$

where $\mathbf{x}$ is the set of segments in the sketch, $\mathbf{c}$ is the set of segment labels, and $Z$ is a normalizing constant.

**Appearance potential.** The appearance potential $\psi_a$ measures how well the candidate group's appearance matches that of its predicted class. It uses the output of the isolated symbol classifier in section 4 and is defined as:

$$\psi_a(c_i, \mathbf{x}) = \log P_a(c_i|\mathbf{x}) \qquad (3)$$

where $P_a(c_i|\mathbf{x})$ is the likelihood score for candidate $c_i$ returned by the isolated symbol classifier.

**Group overlap potential.** The overlap potential $\psi_o(c_i, c_j)$ is a pairwise compatibility that ensures the segment assignments do not conflict with each other. For example, if segments $x_i$ and $x_j$ are both members of candidate $c$ and $x_i$ is assigned to $c$, then $x_j$ must also be assigned to $c$.

$$\psi_o(c_i, c_j) = \begin{cases} -100, & \text{if } ((x_i \in c_j) \text{ or } (x_j \in c_i)) \text{ and } (c_i \neq c_j) \\ 0, & \text{otherwise} \end{cases} \qquad (4)$$

To improve efficiency, instead of connecting every pair of segments that are jointly considered in $c$, we connect the segments into a loop based on temporal ordering. This accomplishes the same constraint with fewer edges. An example is shown in Figure 1(d).

**Joint Context Potential.** The context potential $\psi_c(c_i, c_j, x_i, x_j)$ represents the spatial compatibility between segments $x_i$ and $x_j$, conditioned on their predicted class labels (e.g., resistor-resistor, resistor-wire, etc). It is encoded as a conditional probability table that counts the number of times each spatial relationship ($\theta_1, \theta_2, \theta_3$) occurred for a given class pair (see Figure 3).

$$\psi_c(c_i, c_j, x_i, x_j) = \log P_c(\boldsymbol{\theta}(x_i, x_j) \mid \text{class}(c_i), \text{class}(c_j)) \qquad (5)$$

where $\text{class}(c_i)$ is the predicted class for candidate $c_i$ and $\boldsymbol{\theta}(x_i, x_j)$ is the set of three spatial relationships ($\theta_1, \theta_2, \theta_3$) between segments $x_i$ and $x_j$. This potential is active only for pairs of segments whose distance at the closest point is less than $L/2$. To build the probability table we discretize $\theta_1$ and $\theta_2$ into bins of size $\pi/8$ and $\theta_3$ into bins of size $L/4$.

The entries in the conditional probability table are defined as:

$$P_c(\boldsymbol{\theta} \mid l_i, l_j) = \frac{N_{\theta, \text{class}_i, \text{class}_j} + \alpha}{\sum_{\theta'} N_{\theta', \text{class}_i, \text{class}_j} + \alpha} \qquad (6)$$

where $N_{\theta,\text{class}_i,\text{class}_j}$ is the number of times we observed a pair of segments with spatial relationship $\theta$ and class labels (class$_i$, class$_j$) and $\alpha$ is a weak prior ($\alpha = 10$ in our experiments).

**Inference.** We apply the max-product belief propagation algorithm [22] to find the configuration that maximizes Equation 2. Belief propagation works by iteratively passing messages around the connected nodes in the graph; each message from node $i$ to node $j$ contains $i$'s belief for each possible state of $j$. In our implementation we use an "accelerated" message passing schedule [21] that propagates messages immediately without waiting for other nodes to finish. The procedure alternates between forward and backward passes through the nodes based on the temporal ordering of the segments, running for a total of 100 iterations.

## 5    Evaluation

One goal of our research is to build a system that can handle the range of drawings styles found in natural, real world diagrams. As a result, our data collection program was designed to behave like a piece of paper, i.e., capturing the sketch but providing no recognition or feedback. Using the data we collected, we evaluated five versions of our system:

**Appearance** uses only the isolated appearance-based recognizer from [11].
**Appearance+Geometry** uses isolated appearance and geometric features.
**Appearance+Geometry+Local** uses isolated appearance, geometric features, and local context.
**Complete** is the complete framework described in this paper, using our corner detector.
**Complete** *(corner detector from [15])* is the complete framework, using the corner detector in [15]. (We include this comparison to evaluate the effectiveness of our corner detection algorithm.)

Note that the first three versions still use the group overlap potential to select the best set of consistent candidates.

**Chemistry**
For this evaluation we recruited 10 participants who were familiar with organic chemistry and asked each of them to draw 12 real world organic compounds (e.g., Aspirin, Penicillin, Sildenafil, etc) on a Tablet PC. We performed a set of user-independent performance evaluations, testing our system on one user while using the examples from the other 9 users as training data. By leaving out sketches from the same participant, this evaluation demonstrates how well our system would perform on a new user.

For this domain we noticed that users almost never drew multiple symbols using a single stroke, with the exception of multiple connected straight bonds (e.g., rings). Following this observation, we optimized our candidate extractor to filter out multi-segment candidates that break stroke boundaries.

| Method | Accuracy |
|---|---|
| Complete *(corner detector from [15])* | 0.806 |
| Appearance | 0.889 |
| Appearance+Geometry | 0.947 |
| Appearance+Geometry+Local | 0.958 |
| **Complete** | **0.971** |

Table 1: Overall recognition accuracy for the chemistry dataset.

Note that for this dataset we report only accuracy (recall), because, unlike traditional object detection, there are no overlapping detections and every stroke is assigned to a symbol. Thus, a false positive always causes a false negative, so recall and precision are redundant: e.g., misclassifying one segment in a three-segment "H" makes it impossible to recognize the original "H" correctly.

The results in Table 1 show that our method was able to recognize 97% of the symbols correctly. To be considered a correct recognition, a predicted symbol needs to match both the segmentation and class of the ground truth label. By modeling joint context, the complete framework was able to reduce the error rate by 31% compared to the next best method. Figure 4 (top) shows several sketches interpreted by our system. We can see that the diagrams in this dataset can be very messy,

and exhibit a wide range of drawing styles. Notice that in the center diagram, the system made two errors because the author drew hash bonds differently from all the other users, enclosing them inside a triangle.

**Circuits**

The second dataset is a collection of circuit diagrams collected by Oltmans and Davis [9]. The examples were from 10 users who were experienced in basic circuit design. Each user drew ten or eleven different circuits, and every circuit was required to include a pre-specified set of components.

We again performed a set of user-independent performance evaluations. Because the exact locations of the ground truth labels are somewhat subjective (i.e., it is not obvious whether the resistor label should include the short wire segments on either end), we adopt the same evaluation metric used in the Pascal Challenge [2] and in [9]: a prediction is considered correct if the area of overlap between its bounding box and the ground truth label's bounding box is greater than 50% of the area of their union. Also, since we do not count wire detections for this dataset (as in [9]), we report precision as well as recall.

| Method | Precision | Recall |
|---|---|---|
| *Oltmans 2007 [9]* | 0.257 | 0.739 |
| Complete *(corner detector from [15])* | 0.831 | 0.802 |
| Appearance | 0.710 | 0.824 |
| Appearance+Geometry | 0.774 | 0.832 |
| Appearance+Geometry+Local | 0.879 | 0.874 |
| **Complete** | **0.908** | **0.912** |

Table 2: Overall recognition accuracy for the circuit diagram dataset.

Table 2 shows that our method was able to recognize over 91% of the circuit symbols correctly. Compared to the next best method, the complete framework was able to reduce the error rate by 30%. On this dataset Oltmans and Davis [9] were able to achieve a best recall of 73.9% at a precision of 25.7%. Compared to their reported results, we reduced the error rate by 66% and more than triple the precision. As Figure 4 (bottom) shows, this is a very complicated and messy corpus with significant drawing variations like overtracing and pen drag.

**Runtime**

In the evaluations above, it took on average 0.1 seconds to process a new stroke in the circuits dataset and 0.02 seconds for the chemistry dataset (running on a 3.6 GHz machine, single-thread). With incremental interpretation, the system should be able to easily keep up in real time.

**Related Work**

Sketch recognition is a relatively new field, and we did not find any publicly available benchmarks for the domains we evaluated. In this section, we summarize the performance of existing systems that are similar to ours. Alvarado and Davis [1] proposed using dynamically constructed Bayesian networks to represent the contextual relationships between geometric primitives. They achieved an accuracy of 62% on a circuits dataset similar to ours, but needed to manually segment any strokes that contained more than one symbol. Gennari et al [3] developed a system that searches for symbols in high density regions of the sketch and uses domain knowledge to correct low level recognition errors. They reported an accuracy of 77% on a dataset with 6 types of circuit components. Sezgin and Davis [16] proposed using an HMM to model the temporal patterns of geometric primitives, and reported an accuracy of 87% on a dataset containing 4 types of circuit components.

Shilman et. al. [17] proposed an approach that treats sketch recognition as a visual parsing problem. Our work differs from theirs in that we use a rich model of low-level visual appearance and do not require a pre-defined spatial grammar. Ouyang and Davis [10] developed a sketch recognition system that uses domain knowledge to refine its interpretation. Their work focused on chemical diagrams, and detection was limited to symbols drawn using consecutive strokes. Outside of the sketch recognition community, there is also a great deal of interest in combining appearance and context for problems in computer vision [6, 8, 19].

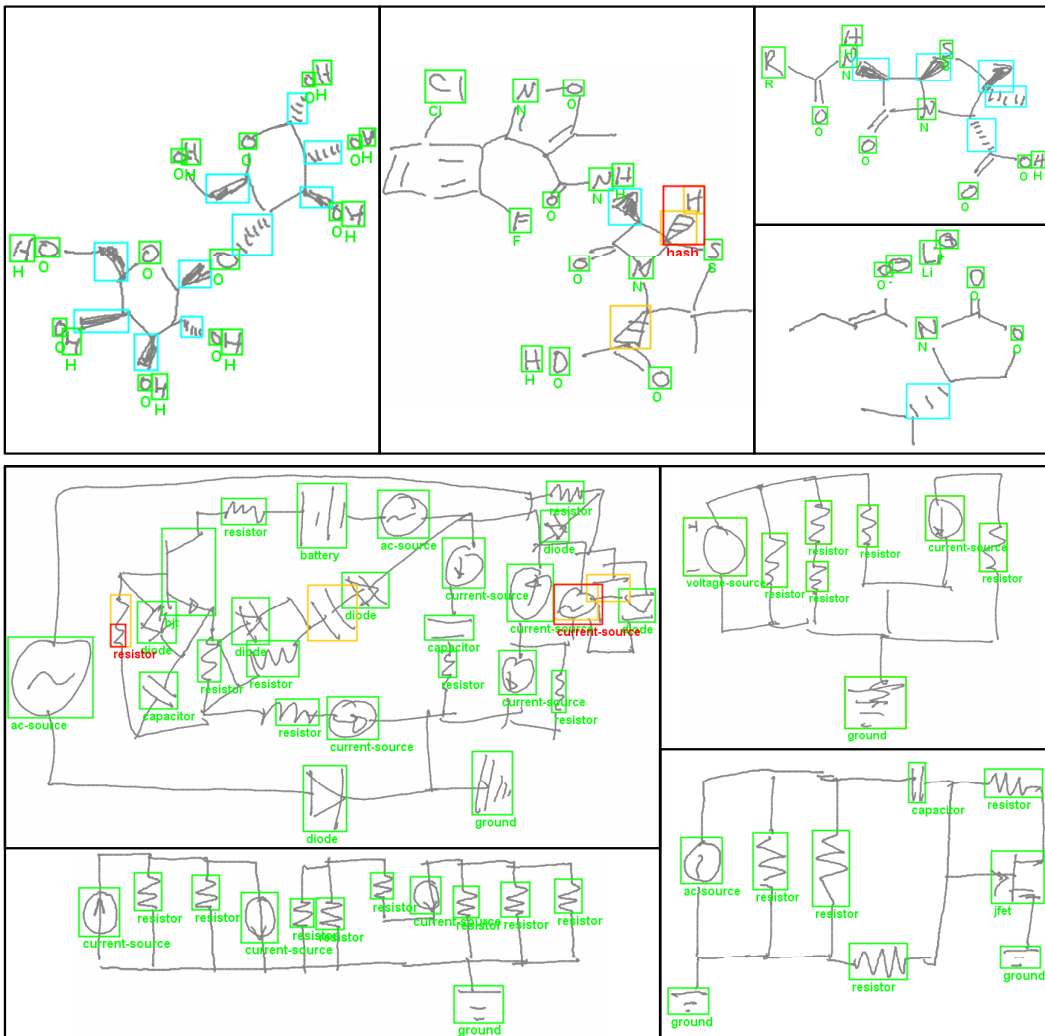

Figure 4: Examples of chemical diagrams (top) and circuit diagrams (bottom) recognized by our system (complete framework). Correct detections are highlighted in green (teal for hash and wedge bonds), false detections in red, and missed symbols in orange.

# 6 Discussion

We have proposed a new framework that combines a rich representation of low level visual appearance with a probabilistic model for capturing higher level relationships. To our knowledge this is the first paper to combine these two approaches, and the result is a recognizer that is better able to handle the range of drawing styles found in messy freehand sketches. To preserve the familiar experience of using pen and paper, our system supports the same symbols, notations, and drawing styles that people are already accustomed to.

In our initial evaluation we apply our method on two real-world domains, chemical diagrams and electrical circuits (with 10 types of components), and achieve accuracy rates of 97% and 91% respectively. Compared to existing benchmarks in literature, our method achieved higher accuracy even though the other systems supported fewer symbols [3, 16], trained on data from the same user [3, 16], or required manual pre-segmentation [1].

**Acknowledgements**
This research was supported in part by a DHS Graduate Research Fellowship and a grant from Pfizer, Inc. We thank Michael Oltmans for kindly making his dataset available to us.

## Footnotes

[1]Defined as the distance between $v_i$ and the line segment formed by $v_{i-1}$ and $v_{i+1}$

[2]In our experiments, we set the threshold to 0.01 times the diagonal length of the stroke's bounding box.

[3]Distance defined as $\mathrm{mindist}(s, g)$ + $\mathrm{bbdist}(s, g)$, where $\mathrm{mindist}(s, g)$ is the distance at the nearest point between segment $s$ and group $g$ and $\mathrm{bbdist}(s, g)$ is the diagonal length of the bounding box containing $s$ and $g$.

[4]The temporal window is 8 strokes for chemistry diagrams and 20 strokes for the circuit diagrams. These parameters were selected empirically, and can be customized by the system designer for each new domain.

# References

[1] C. Alvarado and R. Davis. Sketchread: A multi-domain sketch recognition engine. In *Proc. ACM Symposium on User Interface Software and Technology*, 2004.

[2] M. Everingham, L. Van Gool, C. Williams, J. Winn, and A. Zisserman. The pascal visual object classes challenge 2008 results, 2008.

[3] L. Gennari, L. Kara, T. Stahovich, and K. Shimada. Combining geometry and domain knowledge to interpret hand-drawn diagrams. *Computers & Graphics*, 29(4):547–562, 2005.

[4] M. Gross. The electronic cocktail napkina computational environment for working with design diagrams. *Design Studies*, 17(1):53–69, 1996.

[5] T. Hammond and R. Davis. Ladder: a language to describe drawing, display, and editing in sketch recognition. In *Proc. International Conference on Computer Graphics and Interactive Techniques*, 2006.

[6] X. He, R. Zemel, and M. Carreira-Perpinan. Multiscale conditional random fields for image labeling. In *Proc. IEEE Conference on Computer Vision and Pattern Recognition*, 2004.

[7] L. Kara and T. Stahovich. An image-based, trainable symbol recognizer for hand-drawn sketches. *Computers & Graphics*, 29(4):501–517, 2005.

[8] K. Murphy, A. Torralba, and W. Freeman. Using the forest to see the trees: a graphical model relating features, objects and scenes. *Advances in Neural Information Processing Systems*, 2003.

[9] M. Oltmans. *Envisioning Sketch Recognition: A Local Feature Based Approach to Recognizing Informal Sketches*. PhD thesis, Massachusetts Institute of Technology, Cambridge, MA, May 2007.

[10] T. Y. Ouyang and R. Davis. Recognition of hand drawn chemical diagrams. In *Proc. AAAI Conference on Artificial Intelligence*, 2007.

[11] T. Y. Ouyang and R. Davis. A visual approach to sketched symbol recognition. In *Proc. International Joint Conferences on Artificial Intelligence*, 2009.

[12] J. Platt. Probabilities for sv machines. *Advances in Neural Information Processing Systems*, 1999.

[13] J. Platt. Sequential minimal optimization: A fast algorithm for training support vector machines. *Advances in Kernel Methods-Support Vector Learning*, 1999.

[14] X. Ren and J. Malik. Learning a classification model for segmentation. In *Proc. IEEE International Conference on Computer Vision*, pages 10–17, 2003.

[15] T. Sezgin and R. Davis. Sketch based interfaces: Early processing for sketch understanding. In *Proc. International Conference on Computer Graphics and Interactive Techniques*. ACM New York, NY, USA, 2006.

[16] T. Sezgin and R. Davis. Sketch recognition in interspersed drawings using time-based graphical models. *Computers & Graphics*, 32(5):500–510, 2008.

[17] M. Shilman, H. Pasula, S. Russell, and R. Newton. Statistical visual language models for ink parsing. *Proc. AAAI Spring Symposium on Sketch Understanding*, 2002.

[18] M. Shilman, P. Viola, and K. Chellapilla. Recognition and grouping of handwritten text in diagrams and equations. In *Proc. International Workshop on Frontiers in Handwriting Recognition*, 2004.

[19] J. Shotton, J. Winn, C. Rother, and A. Criminisi. Textonboost: Joint appearance, shape and context modeling for multi-class object recognition and segmentation. *Lecture Notes in Computer Science*, 3951:1, 2006.

[20] M. Szummer. Learning diagram parts with hidden random fields. In *Proc. International Conference on Document Analysis and Recognition*, 2005.

[21] M. Tappen and W. Freeman. Comparison of graph cuts with belief propagation for stereo, using identical mrf parameters. In *Proc. IEEE International Conference on Computer Vision*, 2003.

[22] J. Yedidia, W. Freeman, and Y. Weiss. Understanding belief propagation and its generalizations. *Exploring Artificial Intelligence in the New Millennium*, pages 239–269, 2003.

